# From Regularization Operators to Support Vector Kernels

**Alexander J. Smola**
GMD FIRST
Rudower Chaussee 5
12489 Berlin, Germany
smola@first.gmd.de

**Bernhard Schölkopf**
Max–Planck–Institut für biologische Kybernetik
Spemannstraße 38
72076 Tübingen, Germany
bs@mpik-tueb.mpg.de

## Abstract

We derive the correspondence between regularization operators used in Regularization Networks and Hilbert Schmidt Kernels appearing in Support Vector Machines. More specifically, we prove that the Green's Functions associated with regularization operators are suitable Support Vector Kernels with equivalent regularization properties. As a by–product we show that a large number of Radial Basis Functions namely conditionally positive definite functions may be used as Support Vector kernels.

## 1 INTRODUCTION

Support Vector (SV) Machines for pattern recognition, regression estimation and operator inversion exploit the idea of transforming into a high dimensional feature space where they perform a linear algorithm. Instead of evaluating this map explicitly, one uses Hilbert Schmidt Kernels $k(\mathbf{x}, \mathbf{y})$ which correspond to dot products of the mapped data in high dimensional space, i.e.

$$k(\mathbf{x}, \mathbf{y}) = (\Phi(\mathbf{x}) \cdot \Phi(\mathbf{y})) \tag{1}$$

with $\Phi : \mathbb{R}^n \to \mathcal{F}$ denoting the map into feature space. Mostly, this map and many of its properties are unknown. Even worse, so far no general rule was available which kernel should be used, or why mapping into a very high dimensional space often provides good results, seemingly defying the curse of dimensionality. We will show that each kernel $k(\mathbf{x}, \mathbf{y})$ corresponds to a regularization operator $\hat{P}$, the link being that $k$ is the Green's function of $\hat{P}^*\hat{P}$ (with $\hat{P}^*$ denoting the adjoint operator of $\hat{P}$). For the sake of simplicity we shall only discuss the case of regression — our considerations, however, also hold true for the other cases mentioned above.

We start by briefly reviewing the concept of SV Machines (section 2) and of Regularization Networks (section 3). Section 4 contains the main result stating the equivalence of both

methods. In section 5, we show some applications of this finding to known SV machines. Section 6 introduces a new class of possible SV kernels, and, finally, section 7 concludes the paper with a discussion.

# 2   SUPPORT VECTOR MACHINES

The SV algorithm for regression estimation, as described in [Vapnik, 1995] and [Vapnik et al., 1997], exploits the idea of computing a linear function in high dimensional feature space $\mathcal{F}$ (furnished with a dot product) and thereby computing a nonlinear function in the space of the input data $\mathbb{R}^n$. The functions take the form $f(\mathbf{x}) = (\omega \cdot \Phi(\mathbf{x})) + b$ with $\Phi : \mathbb{R}^n \to \mathcal{F}$ and $\omega \in \mathcal{F}$.

In order to infer $f$ from a training set $\{(\mathbf{x}_i, y_i) \mid i = 1, \dots, \ell, \ \mathbf{x}_i \in \mathbb{R}^n, y_i \in \mathbb{R}\}$, one tries to minimize the empirical risk functional $R_{emp}[f]$ together with a complexity term $\|\omega\|^2$, thereby enforcing *flatness* in feature space, i.e. to minimize

$$R_{reg}[f] = R_{emp}[f] + \lambda \|\omega\|^2 = \frac{1}{\ell} \sum_{i=1}^{\ell} c(f(\mathbf{x}_i), y_i) + \lambda \|\omega\|^2 \tag{2}$$

with $c(f(\mathbf{x}_i), y_i)$ being the cost function determining how deviations of $f(\mathbf{x}_i)$ from the target values $y_i$ should be penalized, and $\lambda$ being a regularization constant. As shown in [Vapnik, 1995] for the case of $\epsilon$–insensitive cost functions,

$$c(f(\mathbf{x}), y) = \begin{cases} |f(\mathbf{x}) - y| - \epsilon & \text{for } |f(\mathbf{x}) - y| \geq \epsilon \\ 0 & \text{otherwise} \end{cases}, \tag{3}$$

(2) can be minimized by solving a quadratic programming problem formulated in terms of dot products in $\mathcal{F}$. It turns out that the solution can be expressed in terms of *Support Vectors*, $\omega = \sum_{i=1}^{\ell} \alpha_i \Phi(\mathbf{x}_i)$, and therefore

$$f(\mathbf{x}) = \sum_{i=1}^{\ell} \alpha_i (\Phi(\mathbf{x}_i) \cdot \Phi(\mathbf{x})) + b = \sum_{i=1}^{\ell} \alpha_i k(\mathbf{x}_i, \mathbf{x}) + b, \tag{4}$$

where $k(\mathbf{x}_i, \mathbf{x})$ is a kernel function computing a dot product in feature space (a concept introduced by Aizerman et al. [1964]). The coefficients $\alpha_i$ can be found by solving a quadratic programming problem (with $K_{ij} := k(\mathbf{x}_i, \mathbf{x}_j)$ and $\alpha_i = \beta_i - \beta_i^*$):

$$\begin{aligned} \text{minimize} \quad & \frac{1}{2} \sum_{i,j=1}^{\ell} (\beta_i^* - \beta_i)(\beta_j^* - \beta_j) K_{ij} - \sum_{i=1}^{\ell} (\beta_i^* - \beta_i) y_i - (\beta_i^* + \beta_i)\epsilon \\ \text{subject to} \quad & \sum_{i=1}^{\ell} \beta_i - \beta_i^* = 0, \quad \beta_i, \beta_i^* \in [0, \tfrac{1}{\lambda \ell}] \end{aligned} \tag{5}$$

Note that (3) is not the only possible choice of cost functions resulting in a quadratic programming problem (in fact quadratic parts and infinities are admissible, too). For a detailed discussion see [Smola and Schölkopf, 1998]. Also note that any continuous symmetric function $k(\mathbf{x}, \mathbf{y}) \in L_2 \otimes L_2$ may be used as an admissible Hilbert–Schmidt kernel if it satisfies Mercer's condition

$$\int \int k(\mathbf{x}, \mathbf{y}) g(\mathbf{x}) g(\mathbf{y}) d\mathbf{x} d\mathbf{y} \geq 0 \quad \text{for all } g \in L_2(\mathbb{R}^n). \tag{6}$$

# 3   REGULARIZATION NETWORKS

Here again we start with minimizing the empirical risk functional $R_{emp}[f]$ plus a regularization term $\|\hat{P}f\|^2$ defined by a regularization operator $\hat{P}$ in the sense of Arsenin and

Tikhonov [1977]. Similar to (2), we minimize

$$R_{reg}[f] = R_{emp} + \lambda \|\hat{P}f\|^2 = \frac{1}{\ell} \sum_{i=1}^{\ell} c(f(\mathbf{x}_i), y_i) + \lambda \|\hat{P}f\|^2. \tag{7}$$

Using an expansion of $f$ in terms of some symmetric function $k(\mathbf{x}_i, \mathbf{x}_j)$ (note here, that $k$ need not fulfil Mercer's condition),

$$f(\mathbf{x}) = \sum_{i} \alpha_i k(\mathbf{x}_i, \mathbf{x}) + b, \tag{8}$$

and the cost function defined in (3), this leads to a quadratic programming problem similar to the one for SVs: by computing Wolfe's dual (for details of the calculations see [Smola and Schölkopf, 1998]), and using

$$D_{ij} := ((\hat{P}k)(\mathbf{x}_i, .) \cdot (\hat{P}k)(\mathbf{x}_j, .)) \tag{9}$$

$((f \cdot g)$ denotes the dot product of the functions $f$ and $g$ in Hilbert Space, i.e. $\int \vec{f}(\mathbf{x})g(\mathbf{x})dx)$, we get $\vec{\alpha} = D^{-1}K(\vec{\beta} - \vec{\beta}^*)$, with $\beta_i, \beta_i^*$ being the solution of

minimize $\quad \frac{1}{2} \sum_{i,j=1}^{\ell} (\beta_i^* - \beta_i)(\beta_j^* - \beta_j)(KD^{-1}K)_{ij} - \sum_{i=1}^{\ell} (\beta_i^* - \beta_i)y_i - (\beta_i^* + \beta_i)\epsilon$

subject to $\quad \sum_{i=1}^{\ell} \beta_i - \beta_i^* = 0, \quad \beta_i, \beta_i^* \in [0, \frac{1}{\ell\lambda}]$

$$\tag{10}$$

Unfortunately this setting of the problem does not preserve sparsity in terms of the coefficients, as a potentially sparse decomposition in terms of $\beta_i$ and $\beta_i^*$ is spoiled by $D^{-1}K$, which in general is not diagonal (the expansion (4) on the other hand does typically have many vanishing coefficients).

## 4 THE EQUIVALENCE OF BOTH METHODS

Comparing (5) with (10) leads to the question if and under which condition the two methods might be equivalent and therefore also under which conditions regularization networks might lead to sparse decompositions (i.e. only a few of the expansion coefficients in $f$ would differ from zero). A sufficient condition is $D = K$ (thus $KD^{-1}K = K$), i.e.

$$k(\mathbf{x}_i, \mathbf{x}_j) = ((\hat{P}k)(\mathbf{x}_i, .) \cdot (\hat{P}k)(\mathbf{x}_j, .)) \tag{11}$$

Our goal now is twofold:

- Given a regularization operator $\hat{P}$, find a kernel $k$ such that a SV machine using $k$ will not only enforce flatness in feature space, but also correspond to minimizing a regularized risk functional with $\hat{P}$ as regularization operator.

- Given a Hilbert Schmidt kernel $k$, find a regularization operator $\hat{P}$ such that a SV machine using this kernel can be viewed as a Regularization Network using $\hat{P}$.

These two problems can be solved by employing the concept of Green's functions as described in [Girosi et al., 1993]. These functions had been introduced in the context of solving differential equations. For our purpose, it is sufficient to know that the Green's functions $G_{\mathbf{x}_i}(\mathbf{x})$ of $\hat{P}^*\hat{P}$ satisfy

$$(\hat{P}^*\hat{P}G_{\mathbf{x}_i})(\mathbf{x}) = \delta_{\mathbf{x}_i}(\mathbf{x}). \tag{12}$$

Here, $\delta_{\mathbf{x}_i}(\mathbf{x})$ is the $\delta$–distribution (not to be confused with the Kronecker symbol $\delta_{ij}$) which has the property that $(f \cdot \delta_{\mathbf{x}_i}) = f(\mathbf{x}_i)$. Moreover we require for all $\mathbf{x}_i$ the projection of $G_{\mathbf{x}_i}(\mathbf{x})$ onto the null space of $\hat{P}^*\hat{P}$ to be zero. The relationship between kernels and regularization operators is formalized in the following proposition.

**Proposition 1**
*Let $\hat{P}$ be a regularization operator, and $G$ be the Green's function of $\hat{P}^*\hat{P}$. Then $G$ is a Hilbert Schmidt–Kernel such that $D = K$. SV machines using $G$ minimize risk functional (7) with $\hat{P}$ as regularization operator.*

**Proof:** *Substituting (12) into* $G_{\mathbf{x}_j}(\mathbf{x}_i) = \left(G_{\mathbf{x}_j}(.) \cdot \delta_{\mathbf{x}_i}(.)\right)$ *yields*

$$G_{\mathbf{x}_j}(\mathbf{x}_i) = \left((\hat{P}G_{\mathbf{x}_i})(.) \cdot (\hat{P}G_{\mathbf{x}_j})(.)\right) = G_{\mathbf{x}_i}(\mathbf{x}_j), \qquad (13)$$

*hence $G(\mathbf{x}_i, \mathbf{x}_j) := G_{\mathbf{x}_i}(\mathbf{x}_j)$ is symmetric and satisfies (11). Thus the SV optimization problem (5) is equivalent to the regularization network counterpart (10). Furthermore $G$ is an admissible positive kernel, as it can be written as a dot product in Hilbert Space, namely*

$$G(\mathbf{x}_i, \mathbf{x}_j) = (\Phi(\mathbf{x}_i) \cdot \Phi(\mathbf{x}_j)) \quad with \quad \Phi : \mathbf{x}_i \longmapsto (\hat{P}G_{\mathbf{x}_i})(.). \qquad (14)$$

In the following we will exploit this relationship in both ways: to compute Green's functions for a given regularization operator $\hat{P}$ and to infer the regularization operator from a given kernel $k$.

# 5   TRANSLATION INVARIANT KERNELS

Let us now more specifically consider regularization operators $\hat{P}$ that may be written as multiplications in Fourier space [Girosi et al., 1993]

$$\left(\hat{P}f \cdot \hat{P}g\right) = \frac{1}{(2\pi)^{n/2}} \int_\Omega \frac{\overline{\tilde{f}(\omega)}\tilde{g}(\omega)}{P(\omega)} d\omega \qquad (15)$$

with $\tilde{f}(\omega)$ denoting the Fourier transform of $f(\mathbf{x})$, and $P(\omega) = P(-\omega)$ real valued, non-negative and converging uniformly to 0 for $|\omega| \to \infty$ and $\Omega = \text{supp}[P(\omega)]$. Small values of $P(\omega)$ correspond to a strong attenuation of the corresponding frequencies.

For regularization operators defined in Fourier Space by (15) it can be shown by exploiting $P(\omega) = P(-\omega) = \overline{P(\omega)}$ that

$$G(\mathbf{x}_i, \mathbf{x}) = \frac{1}{(2\pi)^{n/2}} \int_{\mathbb{R}^n} e^{i\omega(\mathbf{x}_i - \mathbf{x})} P(\omega) d\omega \qquad (16)$$

is a corresponding Green's function satisfying translational invariance, i.e. $G(\mathbf{x}_i, \mathbf{x}_j) = G(\mathbf{x}_i - \mathbf{x}_j)$, and $\tilde{G}(\omega) = P(\omega)$. For the proof, one only has to show that $G$ satisfies (11).

This provides us with an efficient tool for analyzing SV kernels and the types of capacity control they exhibit.

**Example 1 ($B_q$-splines)**
*Vapnik et al. [1997] propose to use $B_q$-splines as building blocks for kernels, i.e.*

$$k(\mathbf{x}) = \prod_{i=1}^{n} B_q(\mathbf{x}_i) \qquad (17)$$

*with $\mathbf{x} \in \mathbb{R}^n$. For the sake of simplicity, we consider the case $n = 1$. Recalling the definition*

$$B_q = \otimes^{q+1} 1_{[-0.5, 0.5]} \qquad (18)$$

*($\otimes$ denotes the convolution and $1_X$ the indicator function on $X$), we can utilize the above result and the Fourier–Plancherel identity to construct the Fourier representation of the corresponding regularization operator. Up to a multiplicative constant, it equals*

$$P(\omega) = \tilde{k}(\omega) = sinc^{(q+1)}\left(\frac{\omega_i}{2}\right). \qquad (19)$$

*This shows that only B-splines of odd order are admissible, as the even ones have negative parts in the Fourier spectrum (which would result in an amplification of the corresponding frequency components). The zeros in $\tilde{k}$ stem from the fact that $B_l$ has only compact support $[-(k+1)/2, (k+1)/2]$. By using this kernel we trade reduced computational complexity in calculating $f$ (we only have to take points with $\|\mathbf{x}_i - \mathbf{x}_j\| \leq c$ from some limited neighborhood determined by $c$ into account) for a possibly worse performance of the regularization operator as it completely removes frequencies $\omega_p$ with $\tilde{k}(\omega_p) = 0$.*

**Example 2 (Dirichlet kernels)**

*In [Vapnik et al., 1997], a class of kernels generating Fourier expansions was introduced,*

$$k(x) = \frac{\sin(2N+1)x/2}{\sin x/2}. \tag{20}$$

*(As in example 1 we consider $\mathbf{x} \in \mathbb{R}^1$ to avoid tedious notation.) By construction, this kernel corresponds to $P(\omega) = \frac{1}{2} \sum_{i=-N}^{N} \delta(\omega - i)$. A regularization operator with these properties, however, may not be desirable as it only damps a finite number of frequencies and leaves all other frequencies unchanged which can lead to overfitting (Fig. 1).*

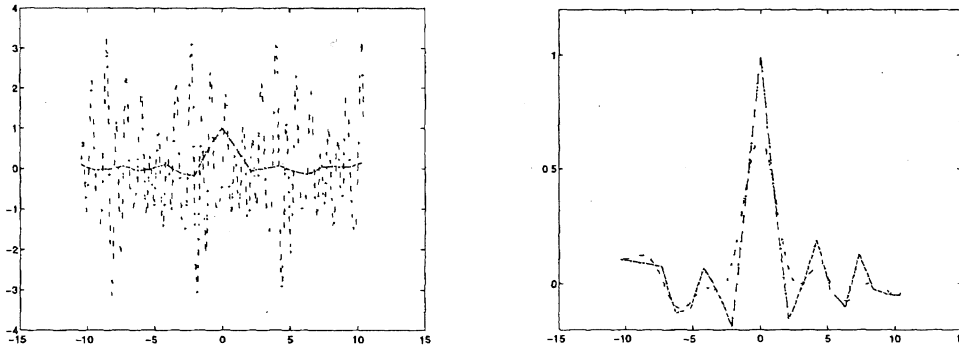

Figure 1: Left: Interpolation with a Dirichlet Kernel of order $N = 10$. One can clearly observe the overfitting (dashed line: interpolation, solid line: original data points, connected by lines). Right: Interpolation of the same data with a Gaussian Kernel of width $\sigma^2 = 1$.

**Example 3 (Gaussian kernels)**

*Following the exposition of Yuille and Grzywacz [1988] as described in [Girosi et al., 1993], one can see that for*

$$\|\hat{P}f\|^2 = \int d\mathbf{x} \sum_m \frac{\sigma^{2m}}{m!2^m} (\hat{O}^m f(\mathbf{x}))^2 \tag{21}$$

*with $\hat{O}^{2m} = \Delta^m$ and $\hat{O}^{2m+1} = \nabla\Delta^m$, $\Delta$ being the Laplacian and $\nabla$ the Gradient operator, we get Gaussians kernels*

$$k(\mathbf{x}) = \exp\left(-\frac{\|\mathbf{x}\|^2}{2\sigma^2}\right). \tag{22}$$

*Moreover, we can provide an equivalent representation of $\hat{P}$ in terms of its Fourier properties, i.e. $P(\omega) = \exp(-\frac{\sigma^2 \|\mathbf{x}\|^2}{2})$ up to a multiplicative constant. Training a SV machine with Gaussian RBF kernels [Schölkopf et al., 1997] corresponds to minimizing the specific cost function with a regularization operator of type (21). This also explains the good performance of SV machines in this case, as it is by no means obvious that choosing a flat function in high dimensional space will correspond to a simple function in low dimensional space, as showed in example 2. Gaussian kernels tend to yield good performance under general smoothness assumptions and should be considered especially if no additional knowledge of the data is available.*

# 6  A NEW CLASS OF SUPPORT VECTOR KERNELS

We will follow the lines of Madych and Nelson [1990] as pointed out by Girosi et al. [1993]. Our main statement is that conditionally positive definite functions (c.p.d.) generate admissible SV kernels. This is very useful as the property of being c.p.d. often is easier to verify than Mercer's condition, especially when combined with the results of Schoenberg and Micchelli on the connection between c.p.d. and completely monotonic functions [Schoenberg, 1938, Micchelli, 1986]. Moreover c.p.d. functions lead to a class of SV kernels that do not necessarily satisfy Mercer's condition.

### Definition 1 (Conditionally positive definite functions)

*A continuous function $h$, defined on $[0, \infty)$, is said to be conditionally positive definite (c.p.d.) of order $m$ on $\mathbb{R}^n$ if for any distinct points $\mathbf{x}_1, \ldots, \mathbf{x}_\ell \in \mathbb{R}^n$ and scalars $c_1, \ldots, c_\ell$ the quadratic form $\sum_{i,j=1}^{\ell} c_i c_j h(\|\mathbf{x}_i - \mathbf{x}_j\|)$ is nonnegative provided that $\sum_{i=1}^{n} c_i p(\mathbf{x}_i) = 0$ for all polynomials $p$ on $\mathbb{R}^n$ of degree lower than $m$.*

### Proposition 2 (c.p.d. functions and admissible kernels)

*Define $\Pi_m^n$ the space of polynomials of degree lower than $m$ on $\mathbb{R}^n$. Every c.p.d. function $h$ of order $m$ generates an admissible Kernel for SV expansions on the space of functions $f$ orthogonal to $\Pi_m^n$ by setting $k(\mathbf{x}_i, \mathbf{x}_j) := h(\|\mathbf{x}_i - \mathbf{x}_j\|^2)$.*

**Proof:** *In [Dyn, 1991] and [Madych and Nelson, 1990] it was shown that c.p.d. functions $h$ generate semi–norms $\|.\|_h$ by*

$$\|f\|_h^2 := \int d\mathbf{x}_i d\mathbf{x}_j h(\|\mathbf{x}_i - \mathbf{x}_j\|) f(\mathbf{x}_i) f(\mathbf{x}_j). \tag{23}$$

*Provided that the projection of $f$ onto the space of polynomials of degree lower than $m$ is zero. For these functions, this, however, also defines a dot product in some feature space. Hence they can be used as SV kernels.*

Only c.p.d. functions of order $m$ up to 2 are of practical interest for SV methods (for details see [Smola and Schölkopf, 1998]). Consequently, we may use kernels like the ones proposed in [Girosi et al., 1993] as SV kernels:

$$k(\mathbf{x}, \mathbf{y}) = \quad e^{-\beta\|\mathbf{x}-\mathbf{y}\|^2} \qquad \text{Gaussian, } (m = 0); \tag{24}$$

$$k(\mathbf{x}, \mathbf{y}) = \quad -\sqrt{\|\mathbf{x} - \mathbf{y}\|^2 + c^2} \qquad \text{multiquadric, } (m = 1) \tag{25}$$

$$k(\mathbf{x}, \mathbf{y}) = \quad \frac{1}{\sqrt{\|\mathbf{x}-\mathbf{y}\|^2+c^2}} \qquad \text{inverse multiquadric, } (m = 0) \tag{26}$$

$$k(\mathbf{x}, \mathbf{y}) = \quad \|\mathbf{x} - \mathbf{y}\|^2 \ln \|\mathbf{x} - \mathbf{y}\| \qquad \text{thin plate splines, } (m = 2) \tag{27}$$

# 7  DISCUSSION

We have pointed out a connection between SV kernels and regularization operators. As one of the possible implications of this result, we hope that it will deepen our understanding of SV machines and of why they have been found to exhibit high generalization ability. In Sec. 5, we have given examples where only the translation into the regularization framework provided insight in why certain kernels are preferable to others. Capacity control is one of the strengths of SV machines; however, this does not mean that the structure of the learning machine, i.e. the choice of a suitable kernel for a given task, should be disregarded. On the contrary, the rather general class of admissible SV kernels should be seen as another strength, provided that we have a means of choosing the right kernel. The newly established link to regularization theory can thus be seen as a tool for constructing the structure consisting of sets of functions in which the SV machine (approximately) performs structural

risk minimization (e.g. [Vapnik, 1995]). For a treatment of SV kernels in a Reproducing Kernel Hilbert Space context see [Girosi, 1997].

Finally one should leverage the theoretical results achieved for regularization operators for a better understanding of SVs (and vice versa). By doing so this theory might serve as a bridge for connecting two (so far) separate threads of machine learning. A trivial example for such a connection would be a Bayesian interpretation of SV machines. In this case the choice of a special kernel can be regarded as a prior on the hypothesis space with $P[f] \propto \exp(-\lambda \|\hat{P}f\|^2)$. A more subtle reasoning probably will be necessary for understanding the capacity bounds [Vapnik, 1995] from a Regularization Network point of view. Future work will include an analysis of the family of polynomial kernels, which perform very well in Pattern Classification [Schölkopf et al., 1995].

## Acknowledgements

AS is supported by a grant of the DFG (# Ja 379/51). BS is supported by the Studienstiftung des deutschen Volkes. The authors thank Chris Burges, Federico Girosi, Leo van Hemmen, Klaus–Robert Müller and Vladimir Vapnik for helpful discussions and comments.

## References

M. A. Aizerman, E. M. Braverman, and L. I. Rozonoér. Theoretical foundations of the potential function method in pattern recognition learning. *Automation and Remote Control*, 25:821–837, 1964.

N. Dyn. Interpolation and approximation by radial and related functions. In C.K. Chui, L.L. Schumaker, and D.J. Ward, editors, *Approximation Theory, VI*, pages 211–234. Academic Press, New York, 1991.

F. Girosi. An equivalence between sparse approximation and support vector machines. A.I. Memo No. 1606, MIT, 1997.

F. Girosi, M. Jones, and T. Poggio. Priors, stabilizers and basis functions: From regularization to radial, tensor and additive splines. A.I. Memo No. 1430, MIT, 1993.

W.R. Madych and S.A. Nelson. Multivariate interpolation and conditionally positive definite functions. II. *Mathematics of Computation*, 54(189):211–230, 1990.

C. A. Micchelli. Interpolation of scattered data: distance matrices and conditionally positive definite functions. *Constructive Approximation*, 2:11–22, 1986.

I.J. Schoenberg. Metric spaces and completely monotone functions. *Ann. of Math.*, 39: 811–841, 1938.

B. Schölkopf, C. Burges, and V. Vapnik. Extracting support data for a given task. In U. M. Fayyad and R. Uthurusamy, editors, *Proc. KDD 1*, Menlo Park, 1995. AAAI Press.

B. Schölkopf, K. Sung, C. Burges, F. Girosi, P. Niyogi, T. Poggio, and V. Vapnik. Comparing support vector machines with gaussian kernels to radial basis function classifiers. *IEEE Trans. Sign. Processing*, 45:2758 – 2765, 1997.

A. J. Smola and B. Schölkopf. On a kernel–based method for pattern recognition, regression, approximation and operator inversion. *Algorithmica*, 1998. see also GMD Technical Report 1997-1064, URL: http://svm.first.gmd.de/papers.html.

V. Vapnik. *The Nature of Statistical Learning Theory*. Springer Verlag, New York, 1995.

V. Vapnik, S. Golowich, and A. Smola. Support vector method for function approximation, regression estimation, and signal processing. In *NIPS 9*, San Mateo, CA, 1997.

A. Yuille and N. Grzywacz. The motion coherence theory. In *Proceedings of the International Conference on Computer Vision*, pages 344–354, Washington, D.C., 1988. IEEE Computer Society Press.

